# An Efficient Method for Gradient-Based Adaptation of Hyperparameters in SVM Models

**S. Sathiya Keerthi**
Yahoo! Research
3333 Empire Avenue
Burbank, CA 91504
selvarak@yahoo-inc.com

**Vikas Sindhwani**
Department of Computer Science
University of Chicago
Chicago, IL 60637
vikass@cs.uchicago.edu

**Olivier Chapelle**
MPI for Biological Cybernetics
Spemannstraße 38
72076 Tübingen
olivier.chapelle@tuebingen.mpg.de

## Abstract

We consider the task of tuning hyperparameters in SVM models based on minimizing a smooth performance validation function, e.g., smoothed k-fold cross-validation error, using non-linear optimization techniques. The key computation in this approach is that of the gradient of the validation function with respect to hyperparameters. We show that for large-scale problems involving a wide choice of kernel-based models and validation functions, this computation can be very efficiently done; often within just a fraction of the training time. Empirical results show that a near-optimal set of hyperparameters can be identified by our approach with very few training rounds and gradient computations.

.

## 1 Introduction

Consider the general SVM classifier model in which, given $n$ training examples $\{(x_i, y_i)\}_{i=1}^n$, the primal problem consists of solving the following problem:

$$\min_{(w,b)} \frac{1}{2}\|w\|^2 + C\sum_{i=1}^n l(o_i, y_i) \tag{1}$$

where $l$ denotes a loss function over labels $y_i \in \{+1, -1\}$ and the outputs $o_i$ on the training set. The machine's output $o$ for any example $x$ is given as $o = w \cdot \phi(x) - b = \sum_{j=1}^n \alpha_j y_j k(x, x_i) - b$ where the $\alpha_i$ are the dual variables, $b$ is the threshold parameter and, as usual, computations involving $\phi$ are handled using the kernel function: $k(x, z) = \phi(x) \cdot \phi(z)$. For example, the Gaussian kernel is given by

$$k(x, z) = \exp(-\gamma\|x - z\|^2) \tag{2}$$

The regularization parameter $C$ and kernel parameters such as $\gamma$ comprise the vector $h$ of hyperparameters in the model. $h$ is usually chosen by optimizing a validation measure (such as the $k$-fold cross validation error) on a grid of values (e.g. a uniform grid in the $(\log C, \log \gamma)$ space). Such a grid search is usually expensive. Particularly, when $n$ is large, this search is so time-consuming that one usually resorts to either default hyperparameter values or crude search strategies. The problem becomes more acute when there are more than two hyperparameters. For example, for feature weighting/selection purposes one may wish to use the following ARD-Gaussian kernel:

$$k(x, z) = \exp(-\sum_t \gamma^t\|x^t - z^t\|^2) \tag{3}$$

where $\gamma^t$ = weight on the $t^{\text{th}}$ feature. In such cases, a grid based search is ruled out. In Figure 1 (see section 5) we show contour plots of performance of an SVM on the $\log C - \log \gamma$ plane for a real-world binary classification problem. These plots show that learning performance behaves "nicely" as

a function of hyperparameters. Intuitively, as $C$ and $\gamma$ are varied one expects the SVM to smoothly transition from providing underfitting solutions to overfitting solutions. Given that this phenomenon seems to occur routinely on real-world learning tasks, a very appealing and principled alternative to grid search is to consider a differentiable version of the performance validation function and invoke non-linear gradient-based optimization techniques for adapting hyperparameters. Such an approach requires the computation of the gradient of the validation function with respect to $h$.

Chapelle et al. (2002) give a number of possibilities for such an approach. One of their most promising methods is to use a differentiable version of the leave-one-out (LOO) error. A major disadvantage of this method is that it requires the expensive computation and storage of the inverse of a kernel sub-matrix corresponding to the support vectors. It is worth noting that, even if, on some large scale problems, the support vector set is of a manageable size at the optimal hyperparameters, the corresponding set can be large when the hyperparameter vector is away from the optimal; on many problems, such a far-off region in the hyperparameter space is usually traversed during the adaptation process!

We highlight the contributions of this paper.
(1) We consider differentiable versions of validation-set-based objective functions for model selection (such as $k$-fold error) and give an efficient method for computing the gradient of this function with respect to $h$. Our method does not require the computation of the inverse of a large kernel sub-matrix. Instead, it only needs a single linear system of equations to be solved, which can be done either by decomposition or conjugate-gradient techniques. In essence, the cost of computing the gradient with respect to $h$ is about the same, and usually much lesser than the cost of solving (1) for a given $h$.
(2) Our method is applicable to a wide range of validation objective functions and SVM models that may involve many hyperparameters. For example, a variety of loss functions can be used together with multiclass classification, regression, structured output or semi-supervised SVM algorithms.
(3) Large-scale empirical results show that with BFGS optimization, just trying about 10-20 hyperparameter points leads to the determination of optimal hyperparameters. Moreover, even as compared to a fine grid search, the gradient procedure provides a more precise placement of hyperparameters leading to better generalization performance. The benefit in terms of efficiency over the grid approach is evident even with just two hyperparameters. We also show the usefulness of our method for tuning more than two hyperparameters when optimizing validation functions such as the $F$ measure and weighted error rate. This is particularly useful for imbalanced problems.

This paper is organized as follows: In section 2, we discuss the general class of SVM models to which our method can be applied. In section 3, we describe our framework and provide the details of the gradient computation for general validation functions. In section 4, we discuss how to develop differentiable versions of several common performance validation functions. Empirical results are presented in section 5. We conclude this paper in section 6. Due to space limitations, several details have been omitted but can be found in the technical report (Keerthi et al. (2006)).

## 2   SVM Classification Models

In this section, we discuss the assumptions required for our method to be applicable. Consider SVM classification models of the form in (1). We assume that *the kernel function $k$ is a continuously differentiable function of $h$*. Three commonly used SVM loss functions are: (1) hinge loss; (2) squared hinge loss; and (3) squared loss. In each of these cases, the solution of (1) is obtained by computing the vector $\alpha$ that solves a dual problem. The solution usually leads to a linear system relating $\alpha$ and $b$:

$$P \left( \begin{array}{c} \alpha \\ b \end{array} \right) = q \tag{4}$$

where $P$ and $q$ are, in general, functions of $h$. We make the following assumption: *Locally around $h$ (at which we are interested in calculating the gradient of the validation function to be defined soon) $P$ and $q$ are continuously differentiable functions of $h$.* We write down $P$ and $q$ for the hinge loss function and discuss the validity of the above assumption. Details for other loss functions are similar.

**Hinge loss.** $l(o_i, y_i) = \max\{0, 1 - y_i o_i\}$. After the solution of (1), the training set indices get partitioned into three sets: $I_0 = \{i : \alpha_i = 0\}$, $I_c = \{i : \alpha_i = C\}$ and $I_u = \{i : 0 < \alpha_i < C\}$. Let $\alpha_0, \alpha_c, \alpha_u, y_c, y_u, e_c, e_u, \Omega_{uc}, \Omega_{uu}$ etc be appropriately defined vectors and matrices. Then (4) is given by

$$\alpha_0 = 0, \quad \alpha_c = Ce_c, \quad \begin{pmatrix} \Omega_{uu} & -y_u \\ -y_u^T & 0 \end{pmatrix} \begin{pmatrix} \alpha_u \\ b \end{pmatrix} = \begin{pmatrix} e_u - \Omega_{uc}\alpha_c \\ y_c^T \alpha_c \end{pmatrix} \tag{5}$$

If the partitions $I_0$, $I_c$ and $I_u$ do not change locally around a given $h$ then assumption 2 holds. Generically, this happens for almost all $h$.

The modified Huber loss function can also be used, though the derivation of (4) for it is more complex than for the three loss functions mentioned above. Recently, weighted hinge loss with asymmetric margins (Grandvalet et al., 2005) has been explored for treating imbalanced problems.

**Weighted Hinge loss.** $l(o_i, y_i) = C_i \max\{0, m_i - y_i o_i\}$. where $C_i = C_+$, $m_i = m_+$ if $y_i = 1$ and $C_i = C_-$, $m_i = m_-$ if $y_i = -1$. Because $C_+$ and $C_-$ are present, the hyperparameter $C$ in (1) can be omitted. The SVM model with weighted hinge loss has four extra hyperparameters, $C_+$, $C_-$, $m_+$ and $m_-$, apart from the kernel hyperparameters. Our methods in this paper allow the possibility of efficiently tuning all these parameters together with kernel parameters.

The method described in this paper is not special to classification models only. It extends to a wide class of kernel methods for which the optimality conditions for minimizing a training objective function can be expressed as a linear system (4) in a continuously differentiable manner[1]. These include many models for multiclass classification, regression, structured output and semi-supervised learning (see Keerthi et al. (2006)).

## 3 The gradient of a validation function

Suppose that for the purpose of hyperparameter tuning, we are given a validation scheme involving a small number of (training set, validation set) partitions, such as: (1) using a single validation set, (2) $k$-fold cross validation, or (3) averaging over $k$ randomly chosen (training set, validation set) partitions. Our method applies to any of these three schemes. To keep notations simple, we explain the ideas only for scheme (1) and expand on the other schemes towards the end of this section. Note that throughout the hyperparameter optimization process, the training-validation splits are fixed.

Let $\{\tilde{x}_l, \tilde{y}_l\}_{l=1}^{\tilde{n}}$ denote the validation set. Let $\tilde{K}_{li} = k(\tilde{x}_l, x_i)$ involving a kernel calculation between an element of a validation set with an element of the training set. The output on the $l^{\text{th}}$ validation example is $\tilde{o}_l = \sum_i \alpha_i y_i \tilde{K}_{li} - b$ which, for convenience, we will rewrite as

$$\tilde{o}_l = \psi_l^T \beta \tag{6}$$

where $\beta$ is a vector containing $\alpha$ and $b$, and $\psi_l$ is a vector containing $y_i \tilde{K}_{li}$, $i = 1, \ldots, n$ and $-1$ as the last element (corresponding to $b$). Let us suppose that the model selection problem is formulated as a non-linear optimization problem:

$$h^\star = \underset{h}{\operatorname{argmin}} f(\tilde{o}_1, \ldots, \tilde{o}_{\tilde{n}}) \tag{7}$$

where $f$ is a differentiable validation function of the outputs $\tilde{o}_l$ which implicitly depend on $h$. In the next section, we will outline the construction of such functions for criteria like error rate, F measure etc. We now discuss the computation of $\nabla_h f$. Let $\theta$ denote a generic parameter in $h$ and let us represent partial derivative of some quantity, say $v$, with respect to $\theta$ as $\dot{v}$. Before writing down expressions for $\dot{f}$, let us discuss how to get $\dot{\beta}$. Differentiating (4) with respect to $\theta$ gives

$$P\dot{\beta} + \dot{P}\beta = \dot{q} \quad \Rightarrow \quad \dot{\beta} = P^{-1}(\dot{q} - \dot{P}\beta) \tag{8}$$

Now let us write down $\dot{f}$.

$$\dot{f} = \sum_{l=1}^{\tilde{n}} (\partial f / \partial \tilde{o}_l) \dot{\tilde{o}}_l \tag{9}$$

where $\dot{\tilde{o}}_l$ is obtained by differentiating (6):

$$\dot{\tilde{o}}_l = \psi_l^T \dot{\beta} + \dot{\psi}_l^T \beta \qquad (10)$$

The computation of $\dot{\beta}$ in (8) is the most expensive step, mainly because it requires $P^{-1}$. Note that, for hinge loss, $P^{-1}$ can be computed in a somewhat cheaper way: only a matrix of the dimension of $I_u$ needs to be inverted. Even then, in large scale problems the dimension of the matrix to be inverted can become so large that even storing it may be a problem; even when large storage is possible, the inverse can be very expensive. Most times, the effective rank of $P$ is much smaller than its dimension. Thus, instead of computing $P^{-1}$ in (8), we can instead solve

$$P\dot{\beta} = (\dot{q} - \dot{P}\beta) \qquad (11)$$

for $\dot{\beta}$ approximately using decomposition methods or iterative methods such as conjugate-gradients. This can improve efficiency as well as take care of memory issues by storing $P$ only partially and computing the remaining parts of $P$ as and when needed. Since the right-hand-side vector $(\dot{q} - \dot{P}\beta)$ in (11) changes for each different $\theta$ with respect to which we are differentiating, we need to solve (11) for each element of $h$. If the number of elements of $h$ is not small (say, we want to use (3) with the MNIST dataset which has more than 700 features) then, even with (11), the computations can still remain very expensive.

We now give a simple trick that shows that if the gradient calculations are re-organized, then *obtaining the solution of just a single linear system suffices for computing the full gradient of $f$ with respect to all elements of $h$.* Let us denote the coefficient of $\dot{\tilde{o}}_l$ in the expression for $\dot{f}$ in (9) by $\delta_l$, i.e.,

$$\delta_l = \partial f / \partial \tilde{o}_l \qquad (12)$$

Using (10) and plugging the expression for $\dot{\beta}$ from (8) into (9) gives

$$\dot{f} = \sum_l \delta_l \dot{\tilde{o}}_l = \sum_l \delta_l (\psi_l^T P^{-1} (\dot{q} - \dot{P}\beta) + \dot{\psi}_l^T \beta) = d^T (\dot{q} - \dot{P}\beta) + \left( \sum_l \delta_l \dot{\psi}_l \right)^T \beta \qquad (13)$$

where $d$ is the solution of

$$P^T d = \left( \sum_l \delta_l \psi_l \right) \qquad (14)$$

The beauty of the reorganization in (13) is that $d$ *is the same* for all variables $\theta$ in $h$ about which the differentiation is being done. Thus (14) needs to be solved only once. In concurrent work (Seeger, 2006) has used a similar idea for kernel logistic regression.

As a word of caution, note that $P$ may not be symmetric. See, e.g., the $P$ arising from (5) for the hinge loss case. Also, the parts corresponding to zero components should be omitted from calculations and the special structure of $P$ should be utilized,e.g., for hinge loss when computing $\dot{P}\beta$ the parts of $\dot{P}$ corresponding to $\alpha_0$ (see (5)) can be ignored. The linear system in the above equation can be efficiently solved using conjugate gradient techniques.

The sequence of steps for the computation of the full gradient of $f$ with respect to $h$ is as follows. First compute $\delta_l$ from (12). For various choices of validation function, we outline this computation in the next section. Then solve (14) for $d$. Then, for each $\theta$ use (13) to get all the derivatives of $f$. The computation of $\dot{P}\beta$ has to be performed for each hyperparameter separately. In problems with many hyperparameters, this is the most expensive part of the gradient computation. Note that in some cases, e.g., $\theta = C$, $\dot{P}\beta$ is immediately obtained. For $\theta = \gamma$ or $\gamma_t$, when using (2,3), one can cache pairwise distance computations while computing the kernel matrix. We have found (see section 5) that the cost of computing the gradient of $f$ with respect to $h$ to be usually much less than the cost of solving (1) and then obtaining $f$.

We can also employ the above ideas in a validation scheme where one uses $k$ training-validation splits (e.g in $k$-fold cross-validation). In this case, for each partition one obtains the linear system (4), corresponding validation outputs (6) and the linear system in (14). The gradient is simply computed by summing over the $k$ partitions, i.e., $\dot{f} = \sum_{j=1}^{k} \dot{f}^{(k)}$ where $\dot{f}^{(k)}$ is given by (13) using the quantities $P, q, d$ etc associated with the $k^{\text{th}}$ partition.

The model selection problem (7) may now be solved using, e.g., Quasi-Newton methods such as BFGS which only require function value and gradient at a hyperparameter setting. In particular,

reaching the minimizer of $f$ too closely is not important. In our implementations we terminate optimization iterations when the following loose termination criterion is met: $|f(h^{k+1}) - f(h^k)| \leq 10^{-3}|f(h^k)|$, where $h^{k+1}$ and $h^k$ are consecutive iterates in the optimization process.

A general concern with descent methods is the presence of local minima. In section 5, we make some encouraging empirical observations in this regard, e.g., local minima problems did not occur for the $C, \gamma$ tuning task; for several other tasks, starting points that work surprisingly well could be easily obtained.

## 4   Smooth validation functions

We consider validation functions that are general functions of the confusion matrix, of the form $f(tp, fp)$ where $tp$ is the number of true positives and $fp$ is the number of false positives. Let $u(z)$ denote the unit step function which is 0 when $z < 0$ and 1 otherwise. Denote $\tilde{u}_l = u(\tilde{y}_l \tilde{o}_l)$, which evaluates to 1 if the $l^{\text{th}}$ example is correctly classified and 0 otherwise. Then, $tp$ and $fp$ can be written as $tp = \sum_{l:\tilde{y}_l=+1} \tilde{u}_l$, $fp = \sum_{l:\tilde{y}_l=-1}(1 - \tilde{u}_l)$. Let $\tilde{n}_+$ and $\tilde{n}_-$ be the number of validation examples in the positive and negative classes. The most commonly used validation function is *error rate*.

**Error rate** ($er$) is simply the percentage of incorrect predictions, i.e., $er = (\tilde{n}_+ - tp + fp)/\tilde{n}$.

For classification problems with imbalanced classes it is usual to consider either *weighted error rate* or a function of *precision* and *recall* such as the *F measure*.

**Weighted Error rate** ($wer$) is given by $wer = (\tilde{n}_+ - tp + \eta fp)/(\tilde{n}_+ + \eta \tilde{n}_-)$, where $\eta$ is the ratio of the cost of misclassifications of the negative class to that of the positive class.

**F measure** ($F$) is the harmonic mean of precision and recall: $F = 2tp/(\tilde{n}_+ + tp + fp)$

Alternatively, one may want to maximize precision under a recall constraint, or maximize the *area under the ROC Curve* or maximize the *precision-recall breakeven point*. See Keerthi et al. (2006) for a discussion on how to treat these cases.

It is common practice to evaluate measures like precision, recall and F measure while varying the threshold on the real-valued classifier output, i.e., at any given threshold $\sigma_0$, $tp$ and $fp$ can be redefined in terms of the following,

$$\tilde{u}_l = u\left(\tilde{y}_l(\tilde{o}_l - \sigma_0)\right) \tag{15}$$

For imbalanced problems one may wish to maximize a score such as the F measure over all values of $\sigma_0$. In such cases, it is appropriate to incorporate $\sigma_0$ as an additional hyperparameter that needs to be tuned. Such bias-shifting is particularly also useful as a compensation mechanism for the mismatch between training objective function and validation function; often one uses an SVM as the underlying classifier even though it is not explicitly trained to minimize the validation function that the practitioner truly cares about. In section 5, we make some empirical observations related to this point.

The validation functions discussed above are based on discrete counts. In order to use gradient-based methods smooth functions of $h$ are needed. To develop smooth versions of validation functions, we define $\tilde{s}_l$, which is a sigmoidal approximation to $\tilde{u}_l$ (15) of the following form:

$$\tilde{s}_l = 1/[1 + \exp\left(-\sigma_1 \tilde{y}_l\left(\tilde{o}_l - \sigma_0\right)\right)] \tag{16}$$

where $\sigma_1 > 0$ is a sigmoidal scale factor. *In general, $\sigma_0, \sigma_1$ may be functions of the validation outputs.* (As discussed above, one may alternatively wish to treat $\sigma_0$ as an additional hyperparameter.) The scale factor $\sigma_1$ influences how closely $\tilde{s}_l$ approximates the step function $\tilde{u}_l$ and hence controls the degree of smoothness in building the sigmoidal approximation. As the hyperparameter space is probed, the magnitude of the outputs can vary quite a bit. $\sigma_1$ takes the scale of the outputs into account. Below we discuss various methods to set $\sigma_0, \sigma_1$.

We build a differentiable version of such a function by simply replacing $\tilde{u}_l$ by $\tilde{s}_l$. Thus, we have $f = f(\tilde{s}_1 \ldots \tilde{s}_{\tilde{n}})$. The value of $\delta_l$ (12) is given by:

$$\delta_l = \frac{\partial f}{\partial \tilde{s}_l}\frac{\partial \tilde{s}_l}{\partial \tilde{o}_l} + \left(\sum_r \frac{\partial f}{\partial \tilde{s}_r}\frac{\partial \tilde{s}_r}{\partial \sigma_0}\right)\frac{\partial \sigma_0}{\partial \tilde{o}_l} + \left(\sum_r \frac{\partial f}{\partial \tilde{s}_r}\frac{\partial \tilde{s}_r}{\partial \sigma_1}\right)\frac{\partial \sigma_1}{\partial \tilde{o}_l} \tag{17}$$

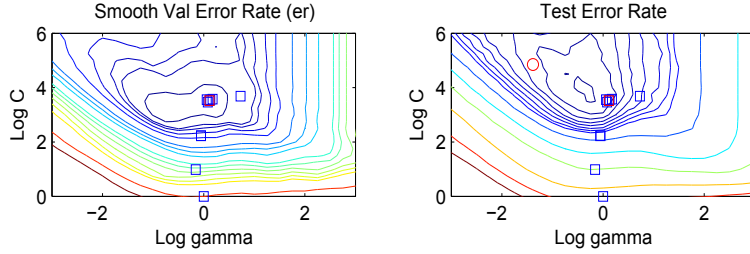

Figure 1: Performance contours for *IJCNN* with 2000 training points. The sequence of points generated by *Grad* are shown by □ (best is in red). The point chosen by *Grid* is shown by ○ in red.

where the partial derivatives of $\tilde{s}_l$ with respect to $\tilde{o}_l, \sigma_0, \sigma_1$ can be easily derived from (16) and $(\partial f/\partial \tilde{s}_l) = (\partial f/\partial tp)(\partial tp/\partial \tilde{s}_l) + (\partial f/\partial fp)(\partial fp/\partial \tilde{s}_l)$.

We now discuss three methods to compute the sigmoidal parameters $\sigma_0, \sigma_1$. For each of these methods the partial derivatives of $\sigma_0, \sigma_1$ with respect to $\tilde{o}_l$ can be obtained (Keerthi et al. (2006)) and used for computing (17).

**Direct Method.** Here, we simply set, $\sigma_0 = 0$, $\sigma_1 = t/\rho$, where $\rho$ denotes standard deviation of the outputs $\{\tilde{o}_l\}$ and $t$ is a constant which is heuristically set to some fixed value in order to well-approximate the step function. In our implementation we use $t = 10$.

**Hyperparameter Bias Method.** Here, we treat $\sigma_0$ as a hyperparameter and set $\sigma_1$ as above.

**Minimization Method.** In this method, we obtain $\sigma_0, \sigma_1$ by performing sigmoidal fitting based on unconstrained minimization of some smooth criterion $N$, i.e., $(\sigma_0, \sigma_1) = \operatorname{argmin}_{\mathcal{R}^2} N$. A natural choice of $N$ is based on Platt's method (Platt (1999)) where $\tilde{s}_l$ is interpreted as the posterior probability that the class of $l^{\text{th}}$ validation example is $\tilde{y}_l$, and we take $N$ to be the *Negative-Log-Likelihood*: $N = N_{nll} = -\sum_l \log(\tilde{s}_l)$. Sigmoidal fitting based on $N_{nll}$ has also been previously proposed in Chapelle et al. (2002). The *probabilistic error rate*: $per = \sum_l (1 - \tilde{s}_l)/\tilde{n}$ and $f = N_{nll}$ are suitable validation functions which go well with the choice $N = N_{nll}$.

## 5 Empirical Results

We demonstrate the effectiveness of our method on several binary classification problems. The SVM model with hinge loss was used. SVM training was done using the SMO algorithm. Five fold cross validation was used to form the validation functions. Four datasets were used: *Adult*, *IJCNN*, *Vehicle* and *Splice*. The first three were taken from *http://www.csie.ntu.edu.tw/~cjlin/libsvmtools/datasets/* and *Splice* was taken from *http://ida.first.fraunhofer.de/~raetsch/*. The number of examples/features in these datasets are: *Adult*: 32561/123; *IJCNN*: 141691/22; *Vehicle*: 98528/100; and *Splice*: 3175/60. For each dataset, training sets of different sizes were chosen in a class-wise stratified fashion; the remaining examples formed the test set.

The Gaussian kernel (2) and the ARD-Gaussian kernel (3) were used. For $(C, \gamma)$ tuning with the Gaussian Kernel, we also tried the popular *Grid* over a $15 \times 15$ grid of values. For $C, \gamma$ tuning with the gradient method, the starting point $C = \gamma = 1$ was used.

**Comparison of validation functions.** Figure 1 shows the contours of the smoothed validation error rate and the actual test error rate for the *IJCNN* dataset with 2000 training examples on the $(\log C, \log \gamma)$ plane. *Grid* and *Grad* respectively denote the grid and the gradient methods applied to the $(C, \gamma)$ tuning task. We used $f = er$ smoothed with the *direct method* for *Grad*. It can be seen that the contours are quite similar. We also generated corresponding contours (omitted) for $f = per$ and $f = N_{nll}$ (see end of section 4) and found that the validation $er$ with the *direct method* better represents the test error rate. Figure 1 also shows that the gradient method very quickly plunges into the high-performance region in the $(C, \gamma)$ space.

**Comparison of Grid and Grad methods.** For various training set sizes of *IJCNN*, in Table 1, we compare the speed and generalization performance of *Grid* and *Grad*, Clearly *Grad* is much more

efficient than *Grid*. The good speed improvement is seen even at small training set sizes. Although the efficiency of *Grid* can be improved in certain ways (say, by performing a crude search followed by a refined search, by avoiding unnecessary exploration of difficult regions in the hyperparameter space etc) *Grad* determines the optimal hyperparameters more precisely. Table 2 compares *Grid* and *Grad* on *Adult* and *Vehicle* datasets for various training sizes. Though the generalization performance of the two methods are close, *Grid* is much slower.

Table 1: Comparison of *Grid*, *Grad* & *Grad-ARD* on *IJCNN* & *Splice*. *nf*= number of hyperparameter vectors tried. (For *Grid*, *nf*= 225.) *cpu*= cpu time in minutes. *erate*=% test error rate.

| | Grid | | Grad | | | Grad-ARD | | |
|---|---|---|---|---|---|---|---|---|
| $n_{\text{trg}}$ | cpu | erate | nf | cpu | erate | nf | cpu | erate |
| *IJCNN* | | | | | | | | |
| 2000 | 10.03 | 2.95 | 11 | 4.58 | 2.87 | 28 | 5.63 | 2.65 |
| 4000 | 38.77 | 2.42 | 12 | 11.40 | 2.42 | 13 | 8.40 | 2.14 |
| 8000 | 218.92 | 1.76 | 14 | 68.58 | 1.77 | 17 | 38.58 | 1.50 |
| 16000 | 1130.37 | 1.24 | 12 | 127.03 | 1.26 | 20 | 154.03 | 1.08 |
| 32000 | 5331.15 | 0.91 | 9 | 382.20 | 0.91 | 7 | 269.16 | 0.82 |
| *Splice* | | | | | | | | |
| 2000 | 11.42 | 9.19 | 13 | 7.57 | 8.17 | 37 | 35.04 | 3.49 |

Table 2: Comparison of *Grad* & *Grid* methods on *Adult* & *Vehicle*. Definitions of *nf*, *cpu* & *erate* are as in Table 1. For *Vehicle* and $n_{\text{trg}}$=16000, *Grid* was discontinued after 5 days of computation.

| | Adult | | | | | Vehicle | | | | |
|---|---|---|---|---|---|---|---|---|---|---|
| | | Grad | | Grid | | | Grad | | Grid | |
| $n_{\text{trg}}$ | nf | cpu | erate | cpu | erate | nf | cpu | erate | cpu | erate |
| 2000 | 9 | 3.62 | 16.21 | 8.66 | 16.14 | 7 | 2.50 | 13.58 | 15.25 | 13.84 |
| 4000 | 16 | 15.98 | 15.64 | 37.53 | 15.95 | 5 | 8.60 | 13.29 | 135.28 | 13.30 |
| 8000 | 10 | 52.17 | 15.69 | 306.25 | 15.59 | 9 | 83.10 | 12.84 | 1458.12 | 12.82 |
| 16000 | 6 | 256.40 | 15.40 | 3667.90 | 15.37 | 6 | 360.88 | 12.58 | – | – |

**Feature Weighting Experiments.** To study the effectiveness of our gradient-based approach when many hyperparameters are present, we use the ARD-Gaussian kernel in (3) and tune $C$ together with all the $\gamma^t$'s. As before, we used $f = er$ smoothed with the *direct method*. The solution for Gaussian kernel was seeded as the starting point for the optimization. Results are reported in Table 1 as *Grad-ARD* where *cpu* denotes the extra time for this optimization. We see that *Grad-ARD* achieves significant improvements in generalization performance over *Grad* without increasing the computational cost by much even though a large number of hyperparameters are being tuned.

**Maximizing F-measure by threshold adjustment.** In section 4 we mentioned about the possible value of threshold adjustment when the validation/test function of interest is a quantity that is different from error rate. We now illustrate this by taking the *Adult* dataset, with *F measure*. The size of the training set is 2000. Gaussian kernel (2) was used. We implemented two methods: (1) we set $\sigma_0 = 0$ and tuned only $C$ and $\gamma$; (2) we tuned the three hyperparameters $C$, $\gamma$ and $\sigma_0$. We ran the methods on ten different random training set/test set splits. Without $\sigma_0$, the mean (standard deviation) of F measure values on 5-fold cross validation and on the test set were: 0.6385 (0.0062) and 0.6363 (0.0081). With $\sigma_0$, the corresponding values improved: 0.6635 (0.0095) and 0.6641 (0.0044). Clearly, the use of $\sigma_0$ yields a very significant improvement on the F-measure. The ability to easily include the threshold as an extra hyperparameter is a very useful advantage for our method.

**Optimizing weighted error rate in imbalanced problems.** In imbalanced problems where the proportion of examples in the positive class is small, one usually minimizes weighted error rate *wer* (see section 4) with a small value of $\eta$. One can think of four possible methods in which, apart from the kernel parameter $\gamma$ and threshold $\sigma_0$ (we used the *Hyperparameter bias method* for smoothening), we include other parameters by considering sub-cases of the weighted hinge loss model (see section 2) – (1) *Usual SVM:* Set $m_+ = m_- = 1$, $C_+ = C$, $C_- = C$ and tune $C$. (2) Set $m_+ = m_- = 1$, $C_+ = C$, $C_- = \eta C$ and tune $C$. (3) Set $m_+ = m_- = 1$ and tune $C_+$ and $C_-$ treating them as independent parameters. (4) Use the full *Weighted Hinge loss* model and tune

$C_+$, $C_-$, $m_+$ and $m_-$. To compare the performance of these methods we took the *IJCNN* dataset, randomly choosing 2000 training examples and keeping the remaining examples as the test set. Ten such random splits were tried. We take $\eta = 0.01$. The top half of Table 3 reports weighted error rates associated with validation and test. The weighted hinge loss model performs best.

Table 3: Mean (standard deviation) of weighted ($\eta = 0.01$) error rate values on the *IJCNN* dataset.

| | $C_+ = C, C_- = C$ | $C_+ = C, C_- = \eta C$ | $C_+, C_-$ tuned | Full Weighted Hinge |
|---|---|---|---|---|
| | *With $\sigma_0$* | | | |
| Validation | 0.0571 (0.0183) | 0.0419 (0.0060) | 0.0490 (0.0104) | 0.0357 (0.0063) |
| Test | 0.0638 (0.0160) | 0.0549 (0.0098) | 0.0571 (0.0136) | 0.0461 (0.0078) |
| | *Without $\sigma_0$* | | | |
| Validation | 0.1953 (0.0557) | 0.1051 (0.0164) | 0.1008 (0.0607) | 0.0364 (0.0061) |
| Test | 0.1861 (0.0540) | 0.0897 (0.0154) | 0.0969 (0.0502) | 0.0469 (0.0076) |

The presence of the threshold parameter $\sigma_0$ is important for the first three methods. The bottom half of Table 3 gives the performance statistics of the methods when threshold is not tuned. Interestingly, for the weighted hinge loss method, tuning of threshold has little effect. Grandvalet et al. (2005) also make the observation that this method appropriately sets the threshold on its own.

**Cost Break-up.** In the gradient-based solution process, each step of the optimization requires the evaluation of $f$ and $\nabla_h f$. In doing this, there are three steps that take up the bulk of the computational cost: (1) training using the SMO algorithm; (2) the solution of the linear system in (14); and (3) the remaining computations associated with the gradient, of which the computation of $\dot{P}\beta$ in (13) is the major part. We studied the relative break-up of the costs for the *IJCNN* dataset (training set sizes ranging from 2000 to 32000), for solution by *Grad* and *Grad-ARD* methods. On an average, the cost of solution by SMO forms 85 to 95% of the total computational time. Thus, the gradient computation is very cheap. We also found that the $\dot{P}\beta$ cost of *Grad-ARD* doesn't become large in spite of the fact that 23 hyperparameters are tuned there. This is mainly due to the efficient reusage of terms in the ARD-Gaussian calculations that we mentioned in section 4.

## 6 Conclusion

The main contribution of this paper is a fast method of computing the gradient of a validation function with respect to hyperparameters for a range of SVM models; together with a nonlinear optimization technique it can be used to efficiently determine the optimal values of many hyperparameters. Even in models with just two hyperparameters our approach is faster and offers a more precise hyperparameter placement than the *Grid* approach. Our approach is particularly of great value for large scale problems. The ability to tune many hyperparameters easily should be used with care. On a text classification problem involving many thousands of features we placed an independent feature weight for each feature and optimized all these weights (together with $C$) only to find severe overfitting taking place. So, for a given problem it is important to choose the set of hyperparameters carefully, in accordance with the richness of the training set.

## Footnotes

[1]Infact, the main ideas easily extend when the optimality conditions form a non-linear system in $(\alpha, b)$ (e.g., in Kernel Logistic Regression).

## References

S. S. Keerthi, V. Sindhwani and O. Chapelle. An efficient method for gradient-based adaptation of hyperparameters in SVM models. Technical Report, 2006.

O. Chapelle, V. Vapnik, O. Bousquet and S. Mukherjee. Choosing multiple parameters for support vector machines. *Machine Learning*, 46:131–159, 2002.

Y. Grandvalet, J. Mariéthoz and S. Bengio. A probabilistic interpretation of SVMs with an application to unbalanced classification. NIPS, 2005.

J. Platt. Probabilities for support vector machines. In *Advances in Large Margin Classifiers*. MIT Press, Cambridge, Massachusetts, 1999.

M. Seeger. Cross validation optimization for structured Hessian kernel methods. Tech. Report, MPI for Biological Cybernetics, Tübingen, Germany, May 2006.
